# Trading off Mistakes and Don't-Know Predictions

**Amin Sayedi**[*]
Tepper School of Business
CMU
Pittsburgh, PA 15213
ssayedir@cmu.edu

**Morteza Zadimoghaddam**[†]
CSAIL
MIT
Cambridge, MA 02139
morteza@mit.edu

**Avrim Blum**[‡]
Department of Computer Science
CMU
Pittsburgh, PA 15213
avrim@cs.cmu.edu

## Abstract

We discuss an online learning framework in which the agent is allowed to say "I don't know" as well as making incorrect predictions on given examples. We analyze the trade off between saying "I don't know" and making mistakes. If the number of don't-know predictions is required to be zero, the model reduces to the well-known mistake-bound model introduced by Littlestone [Lit88]. On the other hand, if no mistakes are allowed, the model reduces to KWIK framework introduced by Li et. al. [LLW08]. We propose a general, though inefficient, algorithm for general finite concept classes that minimizes the number of don't-know predictions subject to a given bound on the number of allowed mistakes. We then present specific polynomial-time algorithms for the concept classes of monotone disjunctions and linear separators with a margin.

## 1 Introduction

Motivated by [KS02, KK99] among others, Li, Littman and Walsh [LLW08] introduced the KWIK framework for online learning, standing for *knows what it knows*. Roughly stated, in the KWIK model, the learning algorithm is required to make only accurate predictions, although it can opt out of predictions by saying "I don't know"($\perp$). After predicting (or answering $\perp$) it is then told the correct answer. The algorithm is not allowed to make any mistakes; still, it learns from those examples on which it answers $\perp$. The goal of the algorithm is to minimize the number of examples on which it answers $\perp$. Several aspects of the model are discussed in [LLW08], and there are many other papers, including [WSDL, DLL09, SL08], using the framework. It is worth mentioning that the idea of forcing the algorithm to say "I don't know" instead of making a mistake has also appeared in earlier work such as [RS88], and referred to as *reliable learning*.

Generally, it is highly desirable to have an algorithm that learns a concept in the KWIK framework using a few, or even polynomial, number of $\perp$s. But unfortunately, for many concepts, no such algorithm exists. In fact, it turns out that even for many basic classes which are very easy to learn in the Mistake-bound model [Lit88], e.g. the class of singletons or disjunctions, the KWIK algorithm needs to say $\perp$ exponentially many times. The purpose of our paper is to relax the assumption of *not making any mistakes*, by allowing a few mistakes, to get much better bounds on the number of $\perp$s. Or, in the other direction, our aim is to produce algorithms that can make substantially fewer mistakes than in the standard Mistake-Bound model, by trading off some of those for (presumably less costly) don't-know predictions.

In [LLW08], the authors show, through a non-polynomial time *enumeration algorithm*, that a finite class $H$ of functions can be learned in the KWIK framework with at most $|H| - 1$ number of $\perp$s.

[*]Part of this work was done when the author was an intern in Microsoft Research New England, MA.

[†]Part of this work was done when the author was an intern in Microsoft Research Cambridge, UK.

[‡]This work was supported in part by NSF grant CCF-0830540.

We show that if only one mistake is allowed, that number can be reduced to $\sqrt{2|H|}$. Furthermore, we show that the problem is equivalent to the famous egg-dropping puzzle, defined formally in Section 2, hence getting bound $(k+1)H^{\frac{1}{k+1}}$ when $k$ mistakes are allowed. Our algorithm does not in general run in polynomial time in the description length of the target function since its running time depends on $|H|$; however, we propose polynomial versions of our algorithm for two important classes: monotone disjunctions and linear separators.

Allowing the algorithm to make mistakes in the KWIK model is equivalent to allowing the algorithm to say "I don't know" in the Mistake-bound model introduced in [Lit88]. In fact, one way of looking at the algorithms presented in section 3 is that we want to decrease the number of mistakes in Mistake-bound model by allowing the algorithm to say $\perp$. The rest of the paper is structured as follows. First we define the model and describe the limits of KWIK model. Then in section 2, we describe how would the bounds on the number of $\perp$s change if we allow a few mistakes in KWIK model. Finally, we give two polynomial algorithms for important classes, Monotone Disjunctions and Linear Separators with a margin, in Section 3.

## 1.1 Model

We want to learn a concept class $H$ consisting of functions $f : X \rightarrow \{+, -\}$. In each stage, the algorithm is given an example $x \in X$ and is asked to predict the target function $h^*(x)$, where we assume $h^* \in H$. The algorithm might answer, $+$, $-$ or $\perp$ representing "I don't know". After the prediction, even if it is $\perp$, the value of $h^*(x)$ is revealed to the algorithm. For a given integer $k$, we want to design an algorithm such that for any sequence of examples, the number of times $M$ that it makes a mistake is not more than $k$, and the number of times $I$ that it answers $\perp$ is minimized.

For example, the special case of $k = 0$ is equivalent to the KWIK framework. Also, if $k \geq \log(|H|)$, the majority vote algorithm can learn the class with no $\perp$ responses, i.e. $I = 0$.

Since we want to derive worst-case bounds, we assume that the sequence of the examples, as well as the target function $h^*$ are selected by an adversary. The adversary sends the examples one by one. For each example $x \in X$, our algorithm decides what to answer; then, the adversary reveals $h^*(x)$.

## 1.2 The KWIK Model

Although the idea of the KWIK framework is quite useful, there are very few problems that can be solved effectively in this framework. The following example demonstrates how an easy problem in the Mistake-bound model can turn into a hard problem in the KWIK model.

**Example 1** *Suppose that $H$ is the class of singletons. In other words, for $h_i \in H$, where $h_i : \{0, 1\}^n \rightarrow \{-, +\}$, we have $h_i(x) = +$ if $x$ is the binary representation of $i$, and $h_i(x) = -$ otherwise. Class $H$ can be learned in Mistake-bound model with mistake bound of only $1$. The algorithm simply predicts $-$ on all examples until it makes a mistake. As soon as the algorithm makes a mistake, it can easily figure out what the target function is.*

*However, class $H$ needs exponentially many $\perp$'s in the KWIK framework to be learned. Since the algorithm does not know the answer until it has seen its first positive example, it must keep answering $\perp$ on all examples that it has not seen yet. Therefore, in the worst case, it answers $\perp$ and finds out that the answer is $-$ on all the first $2^n - 1$ examples that it sees.*

The situation in Example 1 happens for many other classes of functions, e.g. conjunctions or disjunctions, as well.

Next, we review an algorithm (called the *enumeration algorithm* in [LLW08]) for solving problems in the KWIK framework. This algorithm is the main ingredient of most of the algorithms proposed in [LLW08].

**Algorithm 1** *Enumeration*

*The algorithm looks at all the functions in class $H$; if they all agree on the label of the current example $x \in X$, the algorithm outputs that label, otherwise the algorithm outputs $\perp$. Upon receiving the true label of $x$, the algorithm then removes from $H$ those functions $h$ that answered incorrectly*

*on $x$ and continues to the next example. Note that at least one function gets removed from $H$ each time that algorithm answers $\perp$; therefore, the algorithm finds the target function with at most $|H|-1$ number of $\perp$'s.*

## 2 The KWIK Model with Mistakes

Example 1 shows how hard it can be to learn in the KWIK model. To address this, we give the following relaxation of the framework that allows concepts to be learned much more effectively and at the same time preserves the original motivation of the KWIK model—it's better saying "I don't know" rather than making a mistake.

Specifically, we allow the algorithm to make at most $k$ mistakes. Even for very small values of $k$, this can allow us to get much better bounds on the number of times that the algorithm answers $\perp$. For example, by letting $k = 1$, i.e. allowing one mistake, the number of $\perp$'s decreases from $2^n - 1$ to 0 for the class of singletons. Of course, this case is not so interesting since $k = 1$ is the mistake bound for the class. Our main interest is the case that $k > 0$ and yet is much smaller than the number of mistakes needed to learn in the pure Mistake Bound model.

We saw, in Algorithm 1, how to learn a concept class $H$ with no mistakes and with $O(|H|)$ number of $\perp$'s. In many cases, $O(|H|)$ is tight; in fact, if for every subset $S \subseteq H$ with $|S| > 1$ there exists some $x \in X$ for which $|\{h \in S|h(x) = +\}| \in \{1, |S| - 1\}$, then the bound is tight. This condition, for example, is satisfied by the class of intervals: that is, $H = \{[0, a] : a \in \{0, 1, 2, \ldots, 2^n - 1\}\}$.

However, if we allow the algorithm to make one mistake, we show that the number of $\perp$'s can be reduced to $O(\sqrt{|H|})$. In general, if $k$ mistakes are allowed, there is an algorithm that can learn any class $H$ with at most $(k + 1)|H|^{1/k+1}$ number of $\perp$'s. The algorithm is similar to the one for the classic "egg game" puzzle (See [GF]). First suppose that $k = 1$. We make a pool of all functions in $H$, initially consisting of $|H|$ candidates. Whenever an example arrives, we see how many of the candidates label it $+$, and how many label it $-$. If the population of the minority is $< \sqrt{|H|}$, we predict the label that the majority gives on the example; however, if the population of the minority is $\geq \sqrt{|H|}$, we say $\perp$. Those functions that predict incorrectly on an example are removed from the pool in each step; so the pool is just the current version space. If we make a mistake in some step, the size of the version space will reduce to $< \sqrt{|H|}$. Hence, using Algorithm 1, we can complete the learning with at most $\sqrt{|H|}$ number of additional $\perp$'s after our first mistake. Furthermore, note that before making any mistake, we remove at least $\sqrt{|H|}$ of the functions from the pool each time we answer $\perp$. Therefore, the total number of $\perp$'s cannot exceed $2\sqrt{|H|}$. This technique can be generalized for $k$ mistakes, but first we mention a connection between this problem and the classic "egg game" puzzle.

**Example 2** *Egg Game Puzzle*

*You are given 2 identical eggs, and you have access to a $n$-story building. The eggs can be very hard or very fragile or anywhere in between: they may break if dropped from the first floor or may not break even if dropped from the $n$-th floor. You need to figure out the highest floor from which an egg can be dropped without breaking. The question is how many drops you need to make. Note that you can break only two eggs in the process.*

The answer to this puzzle is $\sqrt{2n}$ up to an additive constant. In fact, a thorough analysis of the puzzle when there are $k$ eggs available, instead of just two eggs, is given in [GF]. The $\perp$ minimization problem when $k$ mistakes are allowed is clearly related to the egg game puzzle when the building has $|H|$ floors and there are $k + 1$ eggs available. As a result, with a slightly smarter algorithm that adjusts the threshold $\sqrt{|H|}$ recursively each time an example arrives, we can decrease the number of $\perp$s from $2\sqrt{|H|}$ to $\sqrt{2|H|}$.

**Algorithm 2** *Learning in the KWIK Model with at most $k$ Mistakes*

*Let $s = |H|^{\frac{k}{k+1}}$, and let $P$ denote the current version space: the pool of all functions that might still be the target. Initially $P = H$, but during the learning process, we remove functions from $P$. For each example that arrives, examine how many functions in $P$ label it $+$ and how many label it $-$. If*

*the minority population is $> s$, we answer $\bot$, otherwise, we answer the majority prediction. At the end of each step, we remove the functions that made a mistake in the last step from $P$. Whenever we make a mistake, we update $s = |P|^{\frac{k-i}{k+1-i}}$, where $i$ is the number of mistakes we have made so far.*

**Proposition 1** *Algorithm 2 learns a concept class $H$ with at most $k$ mistakes and $(k+1)|H|^{1/k+1}$ "I don't know"s.*

**Proof:** After the first mistake, the size of the pool reduces to $< |H|^{\frac{k}{k+1}}$. Hence, using induction, we can argue that after the first mistake, the learning can be done with $k-1$ mistakes and $k(|H|^{\frac{k}{k+1}})^{1/k}$ "I don't know"s. There can exist at most $\dfrac{|H|}{|H|^{\frac{k}{k+1}}} = |H|^{1/k+1}$ number of $\bot$'s before the first mistake. Therefore, the total number of $\bot$'s will not exceed

$$|H|^{1/k+1} + k(|H|^{\frac{k}{k+1}})^{1/k} = (k+1)|H|^{1/k+1}.$$

$\square$

Before moving to the next section, we should mention that Algorithm 2 is not computationally efficient. Particularly, if $H$ contains exponentially many functions in the natural parameters of the problem, which is often the case, the running time of Algorithm 2 becomes exponential. In the next section, we give polynomial-time algorithms for two important concept classes.

# 3 The Mistake Bound Model with "I don't know" predictions

We can look at the problem from another perspective: instead of adding mistakes to the KWIK framework, we can add "I don't know" to the Mistake Bound model. In many cases, we prefer our algorithm saying "I don't know" rather than making a mistake. Therefore, in this section, we try to improve over optimal mistake bounds by allowing the algorithm to use a modest number of $\bot$'s, and in general to consider the tradeoff between the number of mistakes and the number of $\bot$'s. Note that an algorithm can always replace its $\bot$'s with random $+$'s and $-$'s, therefore, we must expect that decreasing the number of mistakes by one requires increasing the number of $\bot$'s by at least one.

## 3.1 Monotone Disjunctions

We start with the concept class of *Monotone Disjunctions*. A monotone disjunction is a disjunction in which no literal appears negated, that is, a function of the form

$$f(x_1, \ldots, x_n) = x_{i_1} \vee x_{i_2} \vee \ldots \vee x_{i_k}.$$

Each example is a boolean vector of length $n$, and an example is labeled $+$ if and only if at least one of the variables that belong to the target function is set to 1 in the example. We know that this class can be learned with at most $n$ mistakes in Mistake-bound Model [Lit88] where $n$ is the total number of variables. This class is particularly interesting because results derived about monotone disjunctions can be applied to other classes as well, such as general disjunctions, conjunctions, and $k$-DNF formulas. We are interested in decreasing the number of mistakes at the cost of having (hopefully few) $\bot$'s.

First, let's not worry about the running time and see how well Algorithm 2 performs here. We have $|H| = 2^n$; if we let $k = n/i$, the bound that we get on the number of $\bot$'s will be $\simeq \frac{n2^i}{i}$; this is not bad, especially, for the case of small $i$, e.g. $i = 2, 3$. In fact, for the case of $i = 2$, we are trading off each mistake for four "I don't know"s. But unfortunately, Algorithm 2 cannot do this in polynomial time. Our next goal is to design an algorithm which runs in polynomial time and guarantees the same good bounds on the number of $\bot$'s.

**Algorithm 3** *Learning Monotone Disjunctions with at most $n/2$ Mistakes*

*Let $P$, $P^+$ and $P^-$ be three pools of variables. Initially, $P = \{x_1, \ldots, x_n\}$ and $P^+ = P^- = \phi$. During the process of learning, the variables will be moved from $P$ to $P^-$ or $P^+$. The pool $P^+$ is the set of variables that we know must exist in the target function; the pool $P^-$ is the set of the*

*variables that we know cannot exist in the target function. The learning process finishes by the time that $P$ gets empty.*

*In each step, an example $x$ arrives. Let $S \subseteq \{x_1, \ldots, x_n\}$ be the set representation of $x$, i.e., $x_i \in S$ if and only if $x[i] = 1$. If $S \cap P^+ \neq \phi$, we can say for sure that the example is $+$. If $S \subseteq P^-$, we can say for sure that the example is negative. Otherwise, it must be the case that $S \cap P \neq \phi$, and we cannot be sure about our prediction. Here, if $|S \cap P| \geq 2$ we answer $+$, otherwise, i.e. if $|S \cap P| = 1$, we answer $\perp$.*

*If we make a mistake, we move $S \cap P$ to $P^-$. Every time we answer $\perp$, we move $S \cap P$ to $P^+$ or $P^-$ depending on the correct label of the example.*

**Proposition 2** *Algorithm 3 learns the class of Monotone Disjunctions with at most $M \leq n/2$ mistakes and $n - 2M$ number of $\perp$s.*

**Proof:** If we make a mistake, it must be the case that the answer had been negative while we answered positive; for this to happen, we must have $|S \cap P| \geq 2$. So, after a mistake, we can move $S \cap P$ to $P^-$. The size of $P$, therefore, decreases by at least 2.

Every time we say $\perp$, it must be the case that $|S \cap P| = 1$. Therefore, the label of the example is positive iff $S \cap P$ is contained in the target function, and so the algorithm correctly moves $S \cap P$ to $P^+$ or $P^-$. Additionally, the size of $P$ decreases by at least one on each $\perp$ prediction.  □

Algorithm 3, although very simple, has an interesting property. If in an online learning setting, saying $\perp$ is cheaper than making a mistake, Algorithm 3 strictly dominates the best algorithm in Mistake-bound model. Note that the sum of its $\perp$s and its mistakes is never more than $n$. More precisely, if the cost of making a mistake is 1 and the cost of saying $\perp$ is $< 1$, the worst-case cost of this algorithm is strictly smaller than $n$.

Next we present an algorithm for decreasing the number of mistakes to $n/3$.

**Algorithm 4** *Learning Monotone Disjunctions with at most $n/3$ Mistakes*

*Let $P$, $P^+$, $P^-$ be defined as in Algorithm 3. We have another pool $P'$ which consists of pairs of variables such that for each pair we know at least one of the variables belongs to the target function. As before, the pools form a partition over the set of all variables. In addition, a variable can belong to at most one pair in $P'$. Thus, any given variable is either in a single pair of $P'$ or else in exactly one of the sets $P$, $P^+$, or $P^-$.*

*Whenever an example $x$ arrives we do the following. Let $S \subseteq \{x_1, \ldots, x_n\}$ be the set representation of $x$, i.e. $x_i \in S$ if and only if $x[i] = 1$. If $S \cap P^+ \neq \phi$, we answer $+$. If $S \subseteq P^-$, we answer $-$. Also, if $S$ contains both members of a pair in $P'$, we can say that the label is $+$.*

*If none of the above cases happen, we cannot be sure about our prediction. In this case, if $|S \cap P| \geq 3$, we answer $+$. If $|S \cap (P \cup P')| \geq 2$ and $|S \cap P'| \geq 1$ we again answer $+$. Otherwise, we answer $\perp$. Description of how the algorithm moves variables between sets upon receipt of the correct label is given in the proof below.*

**Proposition 3** *Algorithm 4 learns the class of Monotone Disjunction with at most $M \leq n/3$ mistake and $3n/2 - 3M$ number of $\perp$'s.*

**Proof:** If $|S \cap P| \geq 3$ and we make a mistake on $S$, then the size of $P$ will be reduced by at least 3, and the size of $P^-$ will increase by at least 3. If $|S \cap (P \cup P')| \geq 2$ and $|S \cap P'| \geq 1$ and we make a mistake on $S$, then at least two variables will be moved from $(P' \cup P)$ to $P^-$, and at least one variable will be moved from $P'$ to $P^+$ (since whenever a variable moves from $P'$ to $P^-$, the other variable in its pair should move to $P^+$). Therefore, the size of $P^- \cup P^+$ will increase by at least 3. Since $P^- \cup P^+ \leq n$, we will not make more than $n/3$ mistakes.

There are three cases in which we may answer $\perp$. If $|S \cap P| = 0$ and $|S \cap P'| = 1$, we answer $\perp$; however, after knowing the correct label, $S \cap P'$ will be moved to $P^+$ or $P^-$. Therefore, the number of "I don't know"s of this type is bounded by $n - 3M$. If $|S \cap P| = 1$ and $|S \cap P'| = 0$, again, after knowing the correct label, $S \cap P$ will be moved to $P^+$ or $P^-$, so the same bound applies. If $|S \cap P| = 2$ and $|S \cap P'| = 0$, the correct label might be $+$ or $-$. If it is negative, then we can move

$S \cap P$ to $P^-$ and use the same bound as before. If it is positive, the two variables in $S \cap P$ will be moved to $P'$ as a pair. Note that there can be at most $n/2$ of such $\perp$'s; therefore, the total number of $\perp$'s cannot exceed $n/2 + n - 3M$. □

## 3.2  Learning Linear Separator Functions

In this section, we analyze how we can use $\perp$ predictions to decrease the number of mistakes for efficiently learning linear separators with margin $\gamma$. The high level idea is to use the basic approach of the generic algorithm in Section 2 for finite $H$, but rather than explicitly enumerating over functions, to instead efficiently estimate the measure of the functions in the version space that predict $+$ versus those that predict $-$ and to make prediction decisions based on the result.

**Setting:** We assume a sequence $S$ of $n$ $d$-dimensional examples arrive one by one, and that these examples are linearly separable: there exists a unit-length separator vector $w^*$ such that $w^* \cdot x > 0$ if and only if $x$ is a positive example. Define $\gamma$ to be $min_{x \in S} \frac{|w^* \cdot x|}{|x|}$. For convenience, we will assume that all examples have unit length.

Below, we show how to formulate the problem with a Linear Program to bound the number of mistakes using some "I don't know" answers.

Assume that $n$ examples $x_1, x_2, \cdots, x_n$ are in the sequence $S$. These points arrive one at a time and we have to answer when a point arrives. The objective is to make a small number of mistakes and some "I don't know" answers to find a separation vector $w$ such that $w \cdot x_i$ is positive if and only if $x_i$ is a $+$ point. We can formulate the following linear program using this instance (this sequence of points).

$$w \cdot x_i > 0 \text{ If } x_i \text{ is a } + \text{ instance, and}$$

$$w \cdot x_i \leq 0 \text{ If } x_i \text{ is a } - \text{ instance}$$

Note that there are $d$ variables which are the coordinates of vector $w$, and there are $n$ linear constraints one per input point. Clearly we do not know which points are the $+$ points, so we can not write this linear program explicitly and solve it. But the points arrive one by one and the constraints of this program are revealed over the time. Note that if a vector $w$ is a feasible solution of the above linear program, any positive multiple of $w$ is also a feasible solution. In order to make the analysis easier and bound the core (the set of feasible solutions of the linear program), we can assume that the coordinates of the vector $w$ are always in range $[-1 - \gamma/\sqrt{d}, 1 + \gamma/\sqrt{d}]$. We can add $2d$ linear constraints to make sure that the coordinates do not violate these properties. We will see later why we are choosing the bounds to be $-(1 + \gamma/\sqrt{d})$ and $1 + \gamma/\sqrt{d}$.

Now assume that we are at the beginning and no point has arrived. So we do not have any of the $n$ constraints related to points. The core of the linear program is the set of vectors $w$ in $[-1 - \gamma/\sqrt{d}, 1 + \gamma/\sqrt{d}]^d$ at the beginning. So we have a core (feasible set) of volume $(2 + 2\gamma/\sqrt{d})^d$ at first. For now assume that we can not use the "I don't know" answers. We show how to use them later. The first point arrives. There are two possibilities for this point. It is either a $+$ point or a $-$ point. If we add any of these two constraints to the linear program, we obtain a more restricted linear program with a core of lesser volume. So we obtain one LP for each of these two possibilities, and the sum of the volumes of the cores of these two linear programs is equal to the volume of the core of our current linear program. We will show how to compute these volumes, but for now assume that they are computed. If the volume of the linear program for the $+$ case is larger than the $-$ case, we answer $+$. If our answer is true, we are fine, and we have passed the query with no mistake. Otherwise we have made a mistake, but the volume of the core of our linear program is halved. We do the same for the $-$ case as well, i.e. we answer $-$ when the larger volume is for $-$ case.

Now there are two main issues we have to deal with. First of all, we have to find a way to compute the volume of the core of a linear program. Secondly, we have to find a way to bound the number of mistakes.

In fact computing the volume of a linear program is $\#P$-hard [DF88]. There exists a randomized polynomial time algorithm that approximates the volume of the core of a linear program with $(1+\epsilon)$ approximation [DFK91], i.e. the relative error is $\epsilon$. The running time of this algorithm is polynomial in $n, d$, and $1/\epsilon$. We can use this algorithm to get estimates of the volumes of the linear programs we need. But note that we really do not need to know the volumes of these linear programs. We just need to know whether the volume of the linear program of the $+$ case is larger or the $-$ case is larger or if they are approximately equal. Lovasz and Vempala present a faster polynomial time algorithm for sampling a uniformly random point in the core of a linear program in [LV06]. One way to estimate the relative volumes of both sides is to sample a uniformly random point from the core of our current linear program (without taking into account the new arrived point), and see if the sampled point is in the $+$ side or the $-$ side. If we sample a sufficiently large number of points (here $2\log(n)/\epsilon^2$ is large enough), and if the majority of them are in the $+$ $(-)$ side, we can say that the volume of the linear program for $+$ $(-)$ case is at least a $\frac{1}{2} - \epsilon$ fraction of our current linear program with high probability. So we can answer based on the majority of these sampled points, and if we make a mistake, we know that the volume of the core of the linear program is multiplied by at most $1 - (\frac{1}{2} - \epsilon) = \frac{1}{2} + \epsilon$.

Suppose we have already processed the first $l$ examples and now the $l + 1$st example arrives. We have the linear program with the first $l$ constraints. We sample points from the core of this linear program, and based on the majority of them we answer $+$ or $-$ for this new example. Using the following Theorem, we can bound the number of mistakes.

**Lemma 4** *With high probability $(1 - \frac{1}{n^{\Omega(1)}})$, for every mistake we make in our algorithm, the volume of the core of the linear program decreases by a factor of $(\frac{1}{2} + \epsilon)$.*

**Proof:** Without loss of generality, assume that we answered $+$, but the correct answer was $-$. So we sampled $2\log n/\epsilon^2$ functions uniformly at random from the core, and the majority of them were predicting positive. If less than a $\frac{1}{2} - \epsilon$ fraction of the volume was indeed predicting positive, each sampled point would be from the positive-predicting part with probability less than $\frac{1}{2} - \epsilon$. So the expected number of positive sampled points would be less than $(\frac{1}{2} - \epsilon)(2\log n/\epsilon^2) = (\log n/\epsilon^2 - 2\log n/\epsilon)$. Therefore, by Chernoff bounds, the chance of the sample having a majority of positive-predicting functions would be at most $e^{-(2\log n/\epsilon)^2/2(\log n/\epsilon^2 - 2\log n/\epsilon)} = e^{-2\log n/(1-\epsilon)} = n^{-2/(1-\epsilon)}$. Since there are $n$ examples arriving, we can use the union bound to bound the probability of failure on any of these rounds: the probability that the volume of the core of our linear program is not multiplied by at most $\frac{1}{2} + \epsilon$ on any mistakes is at most $n \times n^{-2/(1-\epsilon)} = \frac{1}{n^{1/(1-\epsilon)}}$. Therefore with high probability (at least $1 - \frac{1}{n^{1/(1-\epsilon)}}$), for every mistake we make, the volume of the core is multiplied by at most $\frac{1}{2} + \epsilon$. $\square$

Now we show that the core of the linear program after adding all $n$ constraints (the constraints of the variables) should have a decent volume in terms of $\gamma$.

**Lemma 5** *If there is a unit-length separator vector $w^*$ with $\min_{x \in S} \frac{w^* \cdot x}{|x|} = \gamma$, the core of the complete linear program after adding all $n$ constraints of the points has volume at least $(\gamma/\sqrt{d})^d$.*

**Proof:** Clearly $w^*$ is in the core of our linear program. Consider a vector $w'$ whose all coordinates are in range $(-\gamma/\sqrt{d}, \gamma/\sqrt{d})$. We claim that $(w^* + w')$ is a correct separator. Consider a point $x_i$. Without loss of generality assume that it is a $+$ point. So $w^* \cdot x_i$ is at least $\gamma \cdot |x_i|$. We also know that $|w' \cdot x_i|$ is at least $-|w'| \cdot |x_i|$. Since all its $d$ coordinates are in range $(-\gamma/\sqrt{d}, \gamma/\sqrt{d})$, we can say that $|w'|$ is less than $\gamma$. So $(w^* + w') \cdot x_i = w^* \cdot x_i + w' \cdot x_i > \gamma|x_i| - \gamma|x_i|$ is positive. We also know that the coordinates of $w^* + w'$ are in range $(-1 - \gamma/\sqrt{d}, 1 + \gamma/\sqrt{d})$ because $w^*$ has unit length (so all its coordinates are between $-1$ and $1$), and the coordinates of $w'$ are in range $(-\gamma/\sqrt{d}, \gamma/\sqrt{d})$. Therefore all vectors of form $w^* + w'$ are in the core. We conclude that the volume of the core is at least $(2\gamma/\sqrt{d})^d$. $\square$

Lemmas 4 and 5 give us the following folklore theorem.

**Theorem 6** *The total number of mistakes in the above algorithm is not more than* $\log_{2/(1+\epsilon)} \frac{(2+2\gamma/\sqrt{d})^d}{(2\gamma/\sqrt{d})^d} = \log_{2/(1+\epsilon)} \frac{(1+\gamma/\sqrt{d})^d}{(\gamma/\sqrt{d})^d} = O(d(\log d + \log 1/\gamma)).$

**Proof:** The proof easily follows from Lemmas 4 and 5. □

Now we make use of the "I don't know" answers to reduce the number of mistakes. Assume that we do not want to make more than $k$ mistakes. Define $Y_1$ to be $(2+2\gamma/\sqrt{d})^d$ which is the volume of the core at the beginning before adding any of the constraints of the points. Define $Y_2$ to be $(2\gamma/\sqrt{d})^d$ which is a lower bound for the volume of the core after adding all the constraints of the points. Let $R$ be the ratio $\frac{Y_2}{Y_1}$. In the above algorithm, we do not make more than $\log_{2/(1+\epsilon)} R$ mistakes.

We want to use "I don't know" answers to reduce this number of mistakes. Define $C$ to be $R^{1/k}$. Let $V, V_1$, and $V_2$ be the volumes of the cores of the current linear program, the linear program with the additional constraint that the new point is a $+$ point, and the linear program with the additional constraint that the new point is a $-$ point respectively. If $V_1/V$ is at most $1/C$, we can say that the new point is a $-$ point. In this case, even if we make a mistake the volume of the core is divided by at least $C$, and by definition of $C$, this can not happen more than $\log_C R = k$ times. Similarly, if $V_2/V$ is at most $1/C$, we can say the new point is a $+$ point. If $V_1/V$ and $V_2/V$ are both greater than $1/C$, we answer "I don't know", and we know that the volume of the core is multiplied by at most $1 - 1/C$.

Since we just need to estimate the ratios $V_1/V$ and $V_2/V$, and in fact we want to see if any of them is smaller than $1/C$ or not, we can simply sample points from the core of our current linear program. But we have to sample at least $O(C \log n)$ points to be able to have reasonable estimates with high probability for these two specific tests (to see if $V_1/V$ or $V_2/V$ is at least $1/C$). For example if we sample $16C \log n$ points, and there are at most $8 \log n$ $+$ points among them, we can say that $V_1/V$ is at most $1/C$ with probability at least $1 - e^{-64 \log^2 n/32 \log n} = 1 - \frac{1}{n^2}$. But if there are at least $8 \log n$ $+$ points, and $8 \log n$ $-$ points among the samples, we can say that both $V_1/V$ and $V_2/V$ are at least $\frac{1}{8C}$ with high probability using Chernoff bounds.

If we make a mistake in this algorithm, the volume of the core is divided by at least $C$, so we do not make more than $k$ mistakes. We also know that for each "I don't know" answer the volume of the core is multiplied by at most $1 - \frac{1}{8C}$, so after $8C$ "I don't know" answers the volume of the core is multiplied by at most $1/e$. Therefore there are at most $O(C \log R)$ "I don't know" answers. This completes the proof of the following theorem.

**Theorem 7** *For any $k > 0$, we can learn a linear separator of margin $\gamma$ in $\Re^d$ using the above algorithm with $k$ mistakes and $O(R^{1/k} \times \log R)$ "I don't know" answers, where $R$ is equal to $\frac{(1+\gamma/\sqrt{d})^d}{(\gamma/\sqrt{d})^d}$.*

## 4   Conclusion

We have discussed a learning framework that combines the elements of the KWIK and mistake-bound models. From one perspective, we are allowing the algorithm to make mistakes in the KWIK model. We showed, using a version-space algorithm and through a reduction to the egg-game puzzle, that allowing a few mistakes in the KWIK model can significantly decrease the number of don't-know predictions.

From another point of view, we are letting the algorithm say "I don't know" in the mistake-bound model. This can be particularly useful if don't-know predictions are cheaper than mistakes and we can trade off some number of mistakes for a not-too-much-larger number of "I don't know"s. We gave polynomial-time algorithms that effectively reduce the number of mistakes in the mistake-bound model using don't-know predictions for two concept classes: monotone disjunctions and linear separators with a margin.

**Acknowledgement**

The authors are very grateful to Adam Kalai, Sham Kakade and Nina Balcan as well as anonymous reviewers for helpful discussions and comments.

## References

[DF88]   Martin E. Dyer and Alan M. Frieze. On the complexity of computing the volume of a polyhedron. *SIAM J. Comput.*, 17(5):967–974, 1988.

[DFK91]  Martin E. Dyer, Alan M. Frieze, and Ravi Kannan. A random polynomial time algorithm for approximating the volume of convex bodies. *J. ACM*, 38(1):1–17, 1991.

[DLL09]  C. Diuk, L. Li, and B.R. Leffler. The adaptive k-meteorologists problem and its application to structure learning and feature selection in reinforcement learning. In *Proceedings of the 26th Annual International Conference on Machine Learning*, pages 249–256. ACM, 2009.

[GF]     Gasarch and Fletcher. The Egg Game. `www.cs.umd.edu/~gasarch/BLOGPAPERS/egg.pdf`.

[KK99]   M. Kearns and D. Koller. Efficient reinforcement learning in factored MDPs. In *International Joint Conference on Artificial Intelligence*, volume 16, pages 740–747. Citeseer, 1999.

[KS02]   M. Kearns and S. Singh. Near-optimal reinforcement learning in polynomial time. *Machine Learning*, 49(2):209–232, 2002.

[Lit88]  N. Littlestone. Learning quickly when irrelevant attributes abound: A new linear-threshold algorithm. *Machine learning*, 2(4):285–318, 1988.

[LLW08]  L. Li, M.L. Littman, and T.J. Walsh. Knows what it knows: a framework for self-aware learning. In *Proceedings of the 25th international conference on Machine learning*, pages 568–575. ACM, 2008.

[LV06]   László Lovász and Santosh Vempala. Hit-and-run from a corner. *SIAM J. Comput.*, 35(4):985–1005, 2006.

[RS88]   R.L. Rivest and R. Sloan. Learning complicated concepts reliably and usefully. In *Proceedings AAAI-88*, pages 635–639, 1988.

[SL08]   A.L. Strehl and M.L. Littman. Online linear regression and its application to model-based reinforcement learning. *Advances in Neural Information Processing Systems*, 20, 2008.

[WSDL]   T.J. Walsh, I. Szita, C. Diuk, and M.L. Littman. Exploring compact reinforcement-learning representations with linear regression. In *Proceedings of the Twenty-Fifth Conference on Uncertainty in Artificial Intelligence (UAI-09), 2009b*.

